# Hierarchical Clustering of a Mixture Model

**Jacob Goldberger    Sam Roweis**
Department of Computer Science, University of Toronto
{jacob,roweis}@cs.toronto.edu

## Abstract

In this paper we propose an efficient algorithm for reducing a large mixture of Gaussians into a smaller mixture while still preserving the component structure of the original model; this is achieved by clustering (grouping) the components. The method minimizes a new, easily computed distance measure between two Gaussian mixtures that can be motivated from a suitable stochastic model and the iterations of the algorithm use only the model parameters, avoiding the need for explicit resampling of datapoints. We demonstrate the method by performing hierarchical clustering of scenery images and handwritten digits.

## 1    Introduction

The Gaussian mixture model (MoG) is a flexible and powerful parametric framework for unsupervised data grouping. Mixture models, however, are often involved in other learning processes whose goals extend beyond simple density estimation to hierarchical clustering, grouping of discrete categories or model simplification. In many such situations we need to group the Gaussians components and re-represent each group by a new single Gaussian density. This grouping results in a compact representation of the original mixture of many Gaussians that respects the original component structure in the sense that no original component is split in the reduced representation. We can view the problem of Gaussian component clustering as general data-point clustering with side information that points belonging to the same original Gaussian component should end up in the same final cluster. Several algorithms that perform clustering of data points given such constraints were recently proposed [11, 5, 12]. In this study we extend these approaches to model-based rather than datapoint based settings. Of course, one could always generate data by sampling from the model, enforcing the constraint that any two samples generated by the same mixture component must end up in the same final cluster. We show that if we already have a parametric representation of the constraint via the MoG density, there is no need for an explicit sampling phase to generate representative datapoints and their associated constraints.

In other situations we want to collapse a MoG into a mixture of fewer components in order to reduce computation complexity. One example is statistical inference in switching dynamic linear models, where performing exact inference with a MoG prior causes the number of Gaussian components representing the current belief to grow exponentially in time. One common solution to this problem is grouping

the Gaussians according to their common history in recent timesteps and collapsing Gaussians grouped together into a single Gaussian [1]. Such a reduction, however, is not based on the parameters of the Gaussians. Other instances in which collapsing MoGs is relevant are variants of particle filtering [10], non-parametric belief propagation [7] and fault detection in dynamical systems [3]. A straight-forward solution for these situations is first to produce samples from the original MoG and then to apply the EM algorithm to learn a reduced model; however this is computationally inefficient and does not preserve the component structure of the original mixture.

## 2   The Clustering Algorithm

We assume that we are given a mixture density $f$ composed of $k$ $d$-dimensional Gaussian components:

$$f(y) = \sum_{i=1}^{k} \alpha_i N(y; \mu_i, \Sigma_i) = \sum_{i=1}^{k} \alpha_i f_i(y) \tag{1}$$

We want to cluster the components of $f$ into a reduced mixture of $m < k$ components. If we denote the set of all ($d$-dimensional) Gaussian mixture models with at most $m$ components by MoG(m), one way to formalize the goal of clustering is to say that we wish to find the element $g$ of MoG(m) "closest" to $f$ under some distance measure. A common proximity criterion is the cross-entropy from $f$ to $g$, i.e. $\hat{g} = \arg\min_g KL(f||g) = \arg\max_g \int f \log g$, where $KL()$ is the Kullback-Leibler divergence and the minimization is performed over all $g$ in MoG(m). This criterion leads to an intractable optimization problem; there is not even a closed-form expression for the KL-divergence between two MoGs let alone an analytic minimizer of its second argument. Furthermore, minimizing a KL-based criterion does not preserving the original component structure of $f$. Instead, we introduce the following new distance measure between $f = \sum_{i=1}^{k} \alpha_i f_i$ and $g = \sum_{j=1}^{m} \beta_j g_j$:

$$d(f, g) = \sum_{i=1}^{k} \alpha_i \min_{j=1}^{m} KL(f_i||g_j) \tag{2}$$

which can be intuitively thought of as the cost of coding data generated by $f$ under the model $g$, if *all points generated by component $i$ of $f$ must be coded under a single component of $g$*. Unlike the KL-divergence between two MoGs, this distance can be analytically computed. In particular, each term is a KL-divergence between two Gaussian distributions $N(\mu_1, \Sigma_1)$ and $N(\mu_2, \Sigma_2)$ which is given by:

$$\frac{1}{2}(\log \frac{|\Sigma_2|}{|\Sigma_1|} + Tr(\Sigma_2^{-1}\Sigma_1) + (\mu_1 - \mu_2)^T \Sigma_2^{-1}(\mu_1 - \mu_2) - d).$$

Under this distance, the optimal reduced MoG representation $\hat{g}$ is the solution to the minimization of (2) over MoG(m): $\hat{g} = \arg\min_g d(f, g)$. Although the minimization ranges over all the MoG(m), we prove that the optimal density $\hat{g}$ is a MoG obtained from grouping the components of $f$ into clusters and collapsing all Gaussians within a cluster into a single Gaussian. There is no closed-form solution for the minimization; rather, we propose an iterative algorithm to obtain a locally optimal solution. Denote the set of all the $m^k$ mappings from $\{1, ..., k\}$ to $\{1, ..., m\}$ by S. For each $\pi \in S$ and $g \in MoG(m)$ define:

$$d(f, g, \pi) = \sum_{i=1}^{k} \alpha_i KL(f_i||g_{\pi(i)}). \tag{3}$$

For a given $g \in MoG(m)$, we associate a matching function $\pi^g \in S$:

$$\pi^g(i) = \arg \min_{j=1}^{m} KL(f_i||g_j) \qquad\qquad i = 1, ..., k \qquad\qquad (4)$$

It can be easily verified that:

$$d(f, g) = d(f, g, \pi^g) = \min_{\pi \in S} d(f, g, \pi) \qquad\qquad (5)$$

i.e. $\pi^g$ is the optimal mapping between the components of $f$ and $g$. Using (5) to define our main optimization we obtain the optimal reduced model as a solution of the following double minimization problem:

$$\hat{g} = \arg \min_g \min_{\pi \in S} d(f, g, \pi) \qquad\qquad (6)$$

For $m > 1$ the double minimization (6) can not be solved analytically. Instead, we can use alternating minimization to obtain a local minimum. Given a matching function $\pi \in S$, we define $g^\pi \in MoG(m)$ as follows. For each $j$ such that $\pi^{-1}(j)$ is non empty, define the following MoG density:

$$f_j^\pi = \frac{\sum_{i \in \pi^{-1}(j)} \alpha_i f_i}{\sum_{i \in \pi^{-1}(j)} \alpha_i} \qquad\qquad (7)$$

The mean and variance of the set $f_j^\pi$, denoted by $\mu'_j$ and $\Sigma'_j$, are:

$$\mu'_j = \frac{1}{\beta_j} \sum_{i \in \pi^{-1}(j)} \alpha_i \mu_i, \qquad \Sigma'_j = \frac{1}{\beta_j} \sum_{i \in \pi^{-1}(j)} \alpha_i \left(\Sigma_i + (\mu_i - \mu'_j)(\mu_i - \mu'_j)^T\right)$$

where $\beta_j = \sum_{i \in \pi^{-1}(j)} \alpha_i$. Let $g_j^\pi = N(\mu'_j, \Sigma'_j)$ be the Gaussian distribution obtained by collapsing the set $f_j^\pi$ into a single Gaussian. It satisfies:

$$g_j^\pi = N(\mu'_j, \Sigma'_j) = \arg \min_g KL(f_j^\pi||g) = \arg \min_g d(f_j^\pi, g)$$

such that the minimization is performed over all the $d$-dimensional Gaussian densities. Denote the collapsed version of $f$ according to $\pi$ by $g^\pi$, i.e.:

$$g^\pi = \sum_{j=1}^{m} \beta_j g_j^\pi \qquad\qquad (8)$$

**Lemma 1:** Given a MoG $f$ and a matching function $\pi \in S$, $g^\pi$ is the unique minimum point of $d(f, g, \pi)$. More precisely, $d(f, g^\pi, \pi) \leq d(f, g, \pi)$ for all $g \in MoG(m)$, and if $d(f, g^\pi, \pi) = d(f, g, \pi)$ then $g_j^\pi = g_j$ for all $j = 1, .., m$ such that $g_j^\pi$ and $g_j$ are the Gaussian components of $g^\pi$ and $g$ respectively.

**Proof:** Denote $c = \sum_{i=1}^{k} \alpha_i \int f_i \log f_i$ (a constant independent of $g$).

$$c - d(f, g, \pi) = \sum_{i=1}^{k} \alpha_i \int f_i \log(g_{\pi(i)}) = \sum_{j=1}^{m} \sum_{i \in \pi^{-1}(j)} \alpha_i \int f_i \log(g_j)$$

$$= \sum_{j=1}^{m} \beta_j \int f_j^\pi \log(g_j) = \sum_{j=1}^{m} \beta_j \int g_j^\pi \log(g_j)$$

The Jensen inequality yields:

$$\leq \sum_{j=1}^{m} \beta_j \int g_j^\pi \log(g_j^\pi) = \sum_{j=1}^{m} \beta_j \int f_j^\pi \log(g_j^\pi) = \sum_{i=1}^{k} \alpha_i \int f_i \log(g_{\pi(i)}^\pi) = c - d(f, g^\pi, \pi)$$

The equality $\int f_j^\pi \log(g_j) = \int g_j^\pi \log(g_j)$ is due to the fact that $\log(g_j)$ is a quadratic expression and the first two moments of $f_j^\pi$ and its collapsed version $g_j^\pi$ are equal. Jensen's inequality is saturated if and only if for all $j = 1, .., m$ (such that $\pi^{-1}(j)$ is not empty) the Gaussian densities $g_j$ and $g_j^\pi$ are equal. $\square$

Using Lemma 1 we obtain a closed form description of a single iteration of the alternating minimization algorithm, which can be viewed as a type of K-means operating at the meta-level of model parameters:

$$
\begin{aligned}
\pi^g &= \arg\min_\pi d(f, g, \pi) & (\textbf{REGROUP}) \\
g^\pi &= \arg\min_g d(f, g, \pi) & (\textbf{REFIT})
\end{aligned}
$$

Above, $\pi^g(i) = \arg\min_j KL(f_i \| g_j)$ and $g^\pi$ is computed using (8). The iterative algorithm monotonically decreases the distance measure $d(f, g)$. Hence, since $S$ is finite, the algorithm converges to a local minimum point after finite number of iterations. The next theorem ensures that once the iterative algorithm converges we obtain a clustering of the MoG components.

**Definition 1**: A MoG $g \in MoG(m)$ is an m-mixture collapsed version of $f$ if there exists a matching function $\pi \in S$ such that $g$ is obtained by collapsing $f$ according to $\pi$, .i.e. $g = g^\pi$.

**Theorem 1:** If applying a single iteration (expressions (REGROUP) and (REFIT)) to a function $g \in MoG(m)$ does not decrease the distance function (2), then necessarily $g$ is a collapsed version of $f$.

**Proof:** Let $g \in MoG(m)$ and let $\pi$ be a matching function such that $d(f, g) = d(f, g, \pi)$. Let $g^\pi$ be a collapsed version of $f$ according to $\pi$. The MoG $g^\pi$ is obtained as a result of applying a single iteration to $g$. Let $g$ be composed of the following Gaussians $\{g_1, ..., g_m\}$ and similarly let $g^\pi = \{g_1^\pi, ..., g_m^\pi\}$. According to Lemma 1, $d(f, g) = d(f, g, \pi) \geq d(f, g^\pi, \pi) \geq d(f, g^\pi)$. Assume that a single iteration does not decrease the distance, i.e. $d(f, g) = d(f, g^\pi)$. Hence $d(f, g, \pi) = d(f, g^\pi, \pi)$. According to Lemma 1, this implies that $g_j = g_j^\pi$ for all $j = 1, ..., m$. Therefore $g$ is a collapsed version of $f$. $\square$

Theorem 1 implies that each local minimum of the propose iterative algorithm is a collapsed version of $f$.

Given the optimal matching function $\pi$, the last step of the algorithm is to set the weights of the reduced representation. $\beta_j^\pi = \sum_{\{i|\pi(i)=j\}} \alpha_i$. These weights are automatically obtained via the collapsing process.

## 3  Experimental Results

In this section we evaluate the performance of our semi-supervised clustering algorithm and compare it to the standard "flat" clustering approach that does not respect the original component structure. We have applied both methods to clustering handwritten digits and natural scene images. In each case, a set of objects is organized in predefined categories. For each category $c$ we learn from a labeled training set a Gaussian distribution $f(x|c)$. A prior distribution over the categories $p(c)$ can be also extracted from the labeled training set. The goal is to cluster the objects into a small number of clusters (fewer than the number of class labels). The standard (flat) approach is to apply an unsupervised clustering to entire collection of original objects, ignoring their class labels. Alternatively we can utilize the given categorization as side-information in order to obtain an improved reduced clustering which also respects the original labels, thus inducing a hierarchical structure.

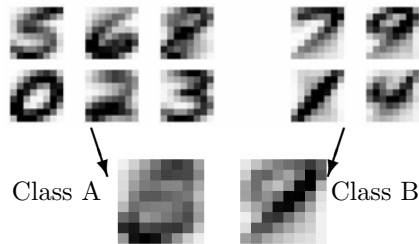

Figure 1: (top) Means of 10 models of digit classes. (bottom) Means of two clusters after our algorithm has grouped 0,2,3,5,6,8 and 1,4,7,9.

Class A    Class B

| method | cls | 0 | 1 | 2 | 3 | 4 | 5 | 6 | 7 | 8 | 9 |
|---|---|---|---|---|---|---|---|---|---|---|---|
| this | Class A | 100 | 4 | 99 | 99 | 3 | 99 | 99 | 0 | 94 | 1 |
| paper | Class B | 0 | 96 | 1 | 1 | 98 | 2 | 1 | 100 | 6 | 99 |
| unsupervised | Class 1 | 93 | 16 | 93 | 87 | 22 | 66 | 96 | 16 | 23 | 25 |
| EM | Class 2 | 7 | 85 | 7 | 14 | 78 | 34 | 4 | 84 | 77 | 76 |

Table 1: Clustering results showing the purity of a 2-cluster reduced model learned from a training set of handwritten digits in 10 original classes. For each true label, the percentage of cases (from an unseen test set) falling into each of the two reduced classes is shown. The top two lines show the purity of assignments provided by our clustering algorithm; the bottom two lines show assignments from a flat unsupervised fitting of a two component mixture.

Our first experiment used a database of handwritten digits. Each example is represented by a $8 \times 8$ grayscale pixel image; 700 cases are used to learn a 64-dimensional full covariance Gaussian distribution for each class. In the next step we want to divide the digits into two natural clusters, while taking into account their original 10-way structure. We applied our semi-supervised algorithm to reduce the mixture of 10 Gaussians into a mixture of two Gaussians. The minimal distance (2) is obtained when the ten digits are divided into the two groups $\{0, 2, 3, 5, 6, 8\}$ and $\{1, 4, 7, 9\}$. The means of the two resulting clusters are shown in Figure 1.

To evaluate the purity of this clustering, the reduced MoG was used to label a test set consists of 4000 previously unseen examples. The binary labels on the test set are obtained by comparing the likelihood of the two components in the reduced mixture. Table 1 (top) presents, for each digit, the percentage of images that were affiliated with each of the two clusters. Alternatively we can apply a standard EM algorithm to learn by maximum likelihood a flat mixture of 2 Gaussians directly from the 7000 training examples, without utilizing their class labels. Table 1 (bottom) shows the results of such an unsupervised clustering, evaluated on the same test set. Although the likelihood of the unsupervised mixture model was significantly better than the semi-supervised model, both on train and test data-sets it is obvious that the purity of the clusters it learns is much worse since it is not preserving the hierarchical class structure. Comparing the top and bottom of Table 1, we can see that using the side information we obtain a clustering of the digit data-base which is much more correlated with categorization of the set into ten digits than the unsupervised procedure.

In a second experiment, we evaluate the performance of our proposed algorithm on image category models. The database used consists of 1460 images selectively hand-picked from the COREL database to create 16 categories. The images within each category have similar color spatial layout, and are labeled with a high-level semantic

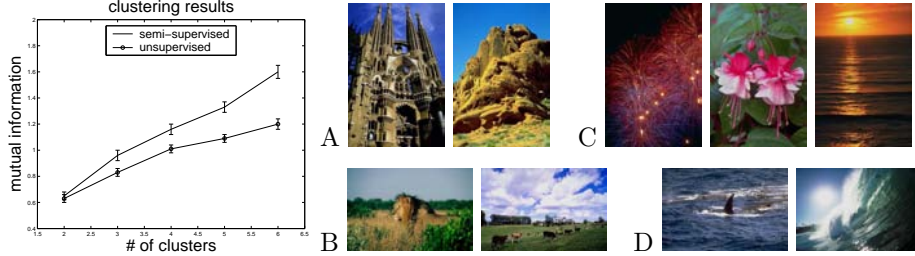

Figure 2: Hierarchical clustering of natural image categories. (left) Mutual information between reduced cluster index and original class. (right) Sample images from the sets A,B,C,D learned by hierarchical clustering.

description (e.g. fields, sunset). For each pixel we extract a five-dimensional feature vector (3 color features and x,y position). From all the pixels that are belonging to the same category we learn a single Gaussian. We have clustered the image categories into $k = 2, ..., 6$ sets using our algorithm and compared the results to unsupervised clustering obtained from an EM procedure that learned a mixture of $k$ Gaussians. In order to evaluate the quality of the clustering in terms of correlation with the category information we computed the mutual information (MI) between the clustering result (into $k$ clusters) and the category affiliation of the images in a test set. A high value of mutual information indicates a strong resemblance between the content of the learned clusters and the hand-picked image categories. It can be verified from the results summarized in Figure 2 that, as we can expect, the MI in the case of semi-supervised clustering is consistently larger than the MI in the case of completely unsupervised clustering. A semi-supervised clustering of the image database yields clusters that are based on both low-level features and a high level available categorization. Sampled images from clustering into 4 sets presented in Figure 2.

## 4   A Stochastic Model for the Proposed Distance

In this section we describe a stochastic process that induces a likelihood function which coincides with the distance measure $d(f, g)$ presented in section 2. Suppose we are given two MoGs:

$$f(y) = \sum_{i=1}^{k} \alpha_i f_i(y) = \sum_{i=1}^{k} \alpha_i N(y; \mu_i, \Sigma_i) \quad , \quad g(y) = \sum_{j=1}^{m} \beta_j g_j(y) = \sum_{j=1}^{m} \beta_j N(y; \mu'_j, \Sigma'_j)$$

Consider an iid sample set of size $n$, drawn from $f(y)$. The samples can be arranged in $k$ blocks according to the Gaussian component that was selected to produce the sample. Assume that $n_i$ samples were drawn from the $i$-th component $f_i$ and denote these samples by $y_i = \{y_{i1}, ..., y_{in_i}\}$. Next, we compute the likelihood of the sample set according to the model $g$; but under the constraint that *samples within the same block must be assigned to the same mixture component of g*. In other words, instead of having a hidden variable for each sample point we shall have one for each sample block. The likelihood of the sample set $y_n$ according to the MoG $g$ under this constraint is:

$$L_n(g) = g(y_1, ..., y_k) = \prod_{i=1}^{k} \sum_{j=1}^{m} \beta_j \prod_{t=1}^{n_i} N(y_{it}; \mu'_j, \Sigma'_j)$$

The main result is that as the number of points sampled grows large, the expected negative log likelihood becomes equal to the distance $d(f,g)$ under the measure proposed above:

**Theorem 2:** For each $g \in MoG(m)$

$$\lim_{n \to \infty} \frac{1}{n} \log L_n(g) = c - d(f,g) \tag{9}$$

such that $c = \sum \alpha_i \int f_i \log f_i$ does not depend on $g$.

Surprisingly, as noted earlier the mixture weights $\beta_j$ do not appear in the asymptotic likelihood function of the generative model presented in this section.

**Proof:** To prove the theorem we shall use the following lemma:

**Lemma 2:** Let $\{x_{jn}\}$ $j = 1,..,m$ be a set of $m$ sequences of real positive numbers such that $x_{jn} \to x_j$ and let $\{\beta_j\}$ be a set of positive numbers. Then $\frac{1}{n} \log \sum_j \beta_j (x_{jn})^n \to \max_j \log x_j$ [This can be shown as follows: Let $a = \arg\max_j x_j$. Then for $n$ sufficiently large, $\beta_a (x_{an})^n \leq \sum_j \beta_j (x_{jn})^n \leq m\beta_a (x_{an})^n$. Hence $\log x_a \leq \lim_{n \to \infty} \frac{1}{n} \log \sum_j \beta_j (x_{jn})^n \leq \log x_a$.]

The points $\{y_{i1}, ..., y_{in_i}\}$ are independently sampled from the Gaussian distribution $f_i$. Therefore, the law of large numbers implies: $\frac{1}{n_i} \log \prod_{t=1}^{n_i} N(y_{it}; \mu'_j, \Sigma'_j) \to \int f_i \log g_j$. Hence, substituting: $x_{jn_i} = (\prod_{t=1}^{n_i} N(y_{it}; \mu'_j, \Sigma'_j))^{\frac{1}{n_i}} \to \exp(\int f_i \log g_j) = x_j$ in Lemma 2, we obtain: $\frac{1}{n_i} \log \sum_{j=1}^{m} \beta_j \prod_{t=1}^{n_i} N(y_{it}; \mu'_j, \Sigma'_j) \to \max_{j=1}^{m} \int f_i \log g_j$ In a similar manner, the law of large numbers, applied to the discrete distribution $(\alpha_1, ..., \alpha_k)$, yields $\frac{n_i}{n} \to \alpha_i$. Hence $\frac{1}{n} \log L_n(g) = \frac{1}{n} \log g(y_1, ..., y_k) = \sum_{i=1}^{k} \frac{n_i}{n} \cdot \frac{1}{n_i} \log \sum_{j=1}^{m} \beta_j \prod_{t=1}^{n_i} N(y_{it}; \mu'_j, \Sigma'_j) \to \sum_{i=1}^{k} \alpha_i \max_{j=1}^{m} \int f_i \log g_j = c - \sum_{i=1}^{k} \alpha_i \min_{j=1}^{m} KL(f_i || g_j) = c - d(f,g)$ $\quad \square$

## 5   Relations to Previous Approaches and Conclusions

Other authors have recently investigated the learning of Gaussian mixture models using various pieces of side information or constraints. Shental et al. [5] utilized the generative model described in the previous section and the EM algorithm derived from it, to learn a MoG from data set endowed with equivalence constraints that enforce equivalent points to be assigned to the same cluster. Vasconcelos and Lippman [9] proposed a similar EM based clustering algorithm for constructing mixture hierarchies using a finite set of virtual samples.

Given the generative model presented above, we can apply the EM algorithm to learn the (locally) maximum likelihood parameters of the reduced MoG model $g(y)$. This EM-based approach, however, is not precisely suitable for our component clustering problem. The EM update rule for the weights of the reduced mixture density is based only on the number of the original components that are clustered into a single component without taking into account the relative weights [9].

The problem discussed in this study is also related to the Information-Bottleneck (IB) principle [8]. In the case of mixture of histograms $f = \sum_{i=1}^{k} \alpha_i f_i$ , the IB principle yields the following iterative algorithm for finding a clustering of a mixture of histograms $g = \sum_{j=1}^{m} \beta_j g_j(y)$:

$$w_{ij} = \frac{\beta_j e^{-\lambda KL(f_i||g_j)}}{\sum_l \beta_l e^{-\lambda KL(f_i||g_l)}} \quad , \quad \beta_j = \sum_i w_{ij} \alpha_i \quad , \quad g_j = \frac{\sum_i w_{ij} \alpha_i f_i}{\sum_i w_{ij} \alpha_i} \tag{10}$$

Assuming that the number of the (virtual) samples tends to $\infty$, we can derive, in a manner similar to the Gaussian case, a grouping algorithm for a mixture of

histograms. Slonim and Weiss [6] showed that the clustering algorithm in this case can be either motivated from the EM algorithm applied to a suitable generative model [4] or from the (hard decision version) of the IB principle [8]. However, when we want to represent the clustering result as a mixture density there *is* a difference in the resulting mixture coefficient between the EM and the IB based algorithms. Unlike the IB updating equation (10) of the coefficients $w_{ij}$ , the EM update equation is based only on the number of components that are collapsed into a single Gaussian. In the case of mixture of Gaussians, applying the IB principle results only in a partitioning of the original components but does not deliver a reduced representation in the form of a smaller mixture [2]. If we modify $g_j$ in equation (10) by collapsing the mixture $g_j$ into a single Gaussian we obtain a soft version of our algorithm. Setting the Lagrange multiplier $\lambda$ to $\infty$ we recover exactly the algorithm described in Section 2.

To conclude, we have presented an efficient Gaussian component clustering algorithm that can be used for object category clustering and for MoG collapsing. We have shown that our method optimizes the distance measure between two MoG that we proposed. In this study we have assumed that the desired number of clusters is given as part of the problem setup, but if this is not the case, standard methods for model selection can be applied.

# References

[1] Y. Bar-Shalom and X. Li. *Estimation and tracking: principles, techniques and software*. Artech House, 1993.

[2] S. Gordon, H. Greenspan, and J. Goldberger. Applying the information bottleneck principle to unsupervised clustering of discrete and continuous image representations. In *ICCV*, 2003.

[3] U. Lerner, R. Parr, D. Koller, and G. Biswas. Bayesian fault detection and diagnosis in dynamic systems. In *AAAI/IAAI, pp. 531–537*, 2000.

[4] J. Puzicha, T. Hofmann, and J. Buhmann. Histogram clustering for unsupervised segmentation and image retrieval. *Pattern Recognition Letters*, 20(9):899–909, 1999.

[5] N. Shental, A. Bar-Hillel, T. Hertz, and D. Weinshall. Computing gaussian mixture models with em using equivalence constraints. In *Proc. of Neural Information Processing Systems*, 2003.

[6] N. Slonim and Y. Weiss. Maximum likelihood and the information bottleneck. In *Proc. of Neural Information Processing Systems*, 2003.

[7] E. Sudderth, A. Ihler, W. Freeman, and A. Wilsky. Non-parametric belief propagation. In *CVPR*, 2003.

[8] N. Tishby, F. Pereira, and W. Bialek. The information bottleneck method. In *Proc. of the 37-th Annual Allerton Conference on Communication, Control and Computing*, pages 368–377, 1999.

[9] N. Vasconcelos and A. Lippman. Learning mixture hierarchies. In *Proc. of Neural Information Processing Systems*, 1998.

[10] J. Vermaak, A. A. Doucet, and P. Perez. Maintaining multi-modality through mixture tracking. In *Int. Conf. on Computer Vision*, 2003.

[11] K. Wagstaff, C. Cardie, S. Rogers, and S. Schroell. Constraind k-means clustering with background knowledge. In *Proc. Int. Conf. on Machine Learning*, 2001.

[12] E.P. Xing, A. Y. Ng, M.I. Jordan, and S. Russell. Distance learning metric. In *Proc. of Neural Information Processing Systems*, 2003.
